# Optimizing Multi-class Spatio-Spectral Filters via Bayes Error Estimation for EEG Classification

**Wenming Zheng**
Research Center for Learning Science
Southeast University
Nanjing, Jiangsu 210096, P.R. China
wenming_zheng@seu.edu.cn

**Zhouchen Lin**
Microsoft Research Asia
Beijing 100190, P.R. China
zhoulin@microsoft.com

## Abstract

The method of common spatio-spectral patterns (CSSPs) is an extension of common spatial patterns (CSPs) by utilizing the technique of delay embedding to alleviate the adverse effects of noises and artifacts on the electroencephalogram (EEG) classification. Although the CSSPs method has shown to be more powerful than the CSPs method in the EEG classification, this method is only suitable for two-class EEG classification problems. In this paper, we generalize the two-class CSSPs method to multi-class cases. To this end, we first develop a novel theory of multi-class Bayes error estimation and then present the multi-class CSSPs (MCSSPs) method based on this Bayes error theoretical framework. By minimizing the estimated closed-form Bayes error, we obtain the optimal spatio-spectral filters of MCSSPs. To demonstrate the effectiveness of the proposed method, we conduct extensive experiments on the BCI competition 2005 data set. The experimental results show that our method significantly outperforms the previous multi-class CSPs (MCSPs) methods in the EEG classification.

## 1 Introduction

The development of non-invasive brain computer interface (BCI) using the electroencephalogram (EEG) signal has become a very hot research topic in the BCI community [1]. During the last several years, a large number of signal processing and machine learning methods have been proposed for EEG classification [6]. It is challenging to extract the discriminant features from the EEG signal for EEG classification. This is because in most cases the EEG data are centered at zero and thus many traditional discriminant feature extraction methods, e.g., Fisher's linear discriminant analysis (FLDA) [7], cannot be successfully used. Among the various EEG feature extraction methods, the common spatial patterns (CSPs) method [2] is one of the most popular. Given two classes of EEG signal, the basic idea of CSPs is to find some projection directions such that the projections of the EEG signal onto these directions will maximize the variance of one class and simultaneously minimize the variance of the other class. Although CSPs have achieved great success in EEG classification, this method only utilizes the spatial information of the EEG signal. To utilize both the spatial and the temporal information of the EEG signal for classification, Lemm et al. [3] proposed a new EEG feature extraction method, called common spatio-spectral patterns (CSSPs), which extended the CSPs method by concatenating the original EEG data and a time-delayed one to form a longer vector sample, and then performed EEG feature extraction, which is similar to the CSPs method, from these padded samples. The experiments in [3] showed that the CSSPs method outperforms the CSPs method.

A multi-class extension of the two-class CSPs method (MCSPs) was proposed by Dornhege et al. [4] who adopted a joint approximate diagonalization (JAD) technique to find the optimal spatial filters. Grosse-Wentrup and Buss [5] recently pointed out that the MCSPs method has two major

drawbacks. The first drawback is that this method lacks solid theoretical foundation with respect to its classification error. The second one is that the selection of the optimal spatial filters of MCSPs is based on heuristics. To overcome these drawbacks, they proposed a method based on mutual information to select the optimal spatial filters from the original MCSPs result. Nevertheless, it should be noted that both the MCSPs methods are based on the JAD technique, where a closed-form solution is unavailable, making the theoretical analysis difficult.

In this paper, we generalize the two-class CSSPs method to multi-class cases, hereafter called the MCSSPs method. However, we do not adopt the same JAD technique used in the MCSPs method to derive our MCSSPs method. Instead, we derive our MCSSPs method directly based on the Bayes error estimation, and thus provide a solid theoretic foundation. To this end, we first develop a novel theory of multi-class Bayes error estimation, which has a closed-form solution to find the optimal discriminant vectors. Based on this new theoretic framework, we propose our MCSSPs method for EEG feature extraction and recognition.

## 2 Brief Review of CSPs and CSSPs

Let $\mathbf{X}_i^t = \{\mathbf{x}_{i,j}^t \in \mathbb{R}^d | j = 1, \cdots, m_{i,t}\}$ $(t = 1, \cdots, n_i; i = 1, \cdots, c)$ denote the EEG data set from the $t$th trial of the $i$th class, where $d$, $c$, $n_i$, and $m_{i,t}$ denote the number of channels (i.e., recording electrodes), the number of classes, the number of trials of the $i$th class, and the number of samples (i.e., recording points) in the $t$th trial of the $i$th class, respectively. Assume that the EEG data conditioned on each class follows a Gaussian distribution with a zero mean, i.e., $p_i(\mathbf{x}) = \mathcal{N}(\mathbf{0}, \mathbf{\Sigma}_i)$ $(i = 1, \cdots, c)$[1]. Then the main task of EEG feature extraction is to find a linear transformation $\mathbf{W} \in \mathbb{R}^{d \times k}$ $(k < d)$, such that for finite training data using the projected vectors $\mathbf{y}_{i,j}^t = \mathbf{W}^T \mathbf{x}_{i,j}^t$ to classify the EEG signal may lead to better classification accuracy than using $\mathbf{x}_{i,j}^t$.

### 2.1 The CSPs Method

For the two-class EEG classification problem, the basic idea of CSPs is to find a transformation matrix $\mathbf{W}$ that simultaneously diagonalizes both class covariance matrices $\mathbf{\Sigma}_1$ and $\mathbf{\Sigma}_2$ [2], i.e.,

$$\mathbf{W}^T \mathbf{\Sigma}_i \mathbf{W} = \mathbf{\Lambda}_i, (i = 1, 2), \tag{1}$$

where $\mathbf{\Lambda}_i = \text{diag}\{\lambda_{i,1}, \cdots, \lambda_{i,d}\}$ $(i = 1, 2)$ are diagonal matrices. The spatial filters can be chosen as the columns of $\mathbf{W}$ associated with the maximal or minimal ratio of $\frac{\lambda_{1,j}}{\lambda_{2,j}}$ $(j = 1, \cdots, d)$. Parra et al. [6] proved that the CSPs method can be formulated as the following optimization problem:

$$\omega = \arg \max_{\omega} \max \left\{ \frac{\omega^T \mathbf{\Sigma}_1 \omega}{\omega^T \mathbf{\Sigma}_2 \omega}, \frac{\omega^T \mathbf{\Sigma}_2 \omega}{\omega^T \mathbf{\Sigma}_1 \omega} \right\}, \tag{2}$$

and this optimization problem boils down to solving the following generalized eigenvalue decomposition problem:

$$\mathbf{\Sigma}_1 \omega = \lambda \mathbf{\Sigma}_2 \omega. \tag{3}$$

Let $\omega_1, \cdots, \omega_d$ and $\lambda_1, \cdots, \lambda_d$ be the eigenvectors and the corresponding eigenvalues of equation (3), then the spatial filters $\omega_{i_1}, \cdots, \omega_{i_k}$ can be chosen from the eigenvectors $\omega_1, \cdots, \omega_d$ associated with the largest and the smallest eigenvalues.

Then $\mathbf{W} = [\omega_{i_1}, \cdots, \omega_{i_k}]$ and the projection of $\mathbf{X}_i^t$ with $\mathbf{W}$ can be expressed as:

$$\mathbf{Y}_i^t = \mathbf{W}^T \mathbf{X}_i^t. \tag{4}$$

### 2.2 The CSSPs Method

The CSSPs method is an extension of CSPs by concatenating the original EEG data and a time-delayed one to form a longer vector sample, and then performing EEG feature extraction, which is similar to the CSPs method, from these padded samples. More specifically, let $\delta^\tau$ denote the time-delay operator with the delayed time $\tau$, i.e.,

$$\delta^\tau(\mathbf{x}_{i,j}^t) = \mathbf{x}_{i,j-\tau}^t. \tag{5}$$

Then, equation (4) can be re-written as the following:

$$\hat{\mathbf{Y}}_i^t = \mathbf{W}_{(0)}^T \mathbf{X}_i^t + \mathbf{W}_{(\tau)}^T \delta^\tau(\mathbf{X}_i^t), \tag{6}$$

where $\mathbf{W}_{(0)}$ and $\mathbf{W}_{(\tau)}$ are the transformation matrices on the EEG data $\mathbf{X}^t$ and $\delta^\tau(\mathbf{X}^t)$, respectively.

To express the above equation in a similar form as CSPs, we define

$$\hat{\mathbf{X}}_i^t = \begin{pmatrix} \mathbf{X}_i^t \\ \delta^\tau(\mathbf{X}_i^t) \end{pmatrix}. \tag{7}$$

In this way, solving the CSSPs problem boils down to solving a similar generalized eigenvalue problem as defined in equation (3), if we use the new class covariance matrices $\hat{\mathbf{\Sigma}}_1$ and $\hat{\mathbf{\Sigma}}_2$ to replace the original class covariance matrices $\mathbf{\Sigma}_1$ and $\mathbf{\Sigma}_2$, where

$$\hat{\mathbf{\Sigma}}_i = \frac{\tilde{\mathbf{\Sigma}}_i}{\text{trace}(\tilde{\mathbf{\Sigma}}_i)}, \quad \text{and} \quad \tilde{\mathbf{\Sigma}}_i = \sum_t \hat{\mathbf{X}}_i^t(\hat{\mathbf{X}}_i^t)^T. \tag{8}$$

## 3    MCSSPs Based on Multi-class Bayes Error Estimation

In this section, we extend the CSSPs method to the multi-class case. To begin with, we develop a novel theory of multi-class Bayes error estimation. Then we present our MCSSPs method based on this Bayes error framework.

### 3.1    Multi-class Bayes Error Estimation

It is well known that the Bayes error regarding classes $i$ and $j$ can be expressed as [7]:

$$\varepsilon = \int \min(P_i p_i(\mathbf{x}), P_j p_j(\mathbf{x}))d\mathbf{x}, \tag{9}$$

where $P_i$ and $p_i(\mathbf{x})$ are the apriori probability and the probability density function of the $i$th class, respectively. Let $\varepsilon_{ij} = \int \sqrt{P_i P_j p_i(\mathbf{x})p_j(\mathbf{x})}d\mathbf{x}$. By applying the following inequality:

$$\min(a,b) \leq \sqrt{ab}, \quad \forall a,b \geq 0, \tag{10}$$

and the assumption $p_i(\mathbf{x}) = \mathcal{N}(\mathbf{0}, \mathbf{\Sigma}_i)$, we obtain the following upper bound of the Bayes error:

$$\varepsilon \leq \varepsilon_{ij} = \sqrt{P_i P_j} \exp\left(-\frac{1}{2}\ln\frac{|\bar{\mathbf{\Sigma}}_{ij}|}{\sqrt{|\mathbf{\Sigma}_i||\mathbf{\Sigma}_j|}}\right) = \sqrt{P_i P_j}\left(\frac{|\bar{\mathbf{\Sigma}}_{ij}|}{\sqrt{|\mathbf{\Sigma}_i||\mathbf{\Sigma}_j|}}\right)^{-\frac{1}{2}}, \tag{11}$$

where $\bar{\mathbf{\Sigma}}_{ij} = \frac{1}{2}(\mathbf{\Sigma}_i + \mathbf{\Sigma}_j)$. The expression in $\exp(\cdot)$ is the simplified Bhattacharyya distance [7]. If we project the samples to 1D by a vector $\omega$, then the upper bound $\varepsilon_{ij}$ becomes:

$$\varepsilon_{ij} = \sqrt{P_i P_j}\left(\frac{\omega^T \bar{\mathbf{\Sigma}}_{ij}\omega}{\sqrt{(\omega^T \mathbf{\Sigma}_i \omega)(\omega^T \mathbf{\Sigma}_j \omega)}}\right)^{-\frac{1}{2}}. \tag{12}$$

Define $u = \omega^T \bar{\mathbf{\Sigma}}_{ij}\omega$ and $v = \omega^T \Delta\mathbf{\Sigma}_{ij}\omega$, where $\Delta\mathbf{\Sigma}_{ij} = \frac{1}{2}(\mathbf{\Sigma}_i - \mathbf{\Sigma}_j)$. Then $\varepsilon_{ij}$ can be written as

$$\varepsilon_{ij} = \sqrt{P_i P_j}\left(\frac{u}{\sqrt{u^2 - v^2}}\right)^{-\frac{1}{2}} = \sqrt{P_i P_j}\left(1 - \left(\frac{v}{u}\right)^2\right)^{\frac{1}{4}} \leq \sqrt{P_i P_j}\left(1 - \frac{1}{4}\left(\frac{v}{u}\right)^2\right). \tag{13}$$

For the $c$ classes problem, the upper bound of the Bayes error in the reduced feature space can be estimated as $\varepsilon \leq \sum_{i=1}^{c-1}\sum_{j=i+1}^{c}\varepsilon_{ij}$ [8]. Then, from equation (13), we obtain that

$$\begin{aligned}
\varepsilon &\leq \sum_{i=1}^{c-1}\sum_{j=i+1}^{c}\varepsilon_{ij} \leq \sum_{i=1}^{c-1}\sum_{j=i+1}^{c}\sqrt{P_i P_j}\left(1 - \frac{1}{4}\left(\frac{\omega^T \Delta\mathbf{\Sigma}_{ij}\omega}{\omega^T \bar{\mathbf{\Sigma}}_{ij}\omega}\right)^2\right) \\
&= \sum_{i=1}^{c-1}\sum_{j=i+1}^{c}\sqrt{P_i P_j} - \frac{1}{8}\sum_{i=1}^{c}\sum_{j=1}^{c}\sqrt{P_i P_j}\left(\frac{\omega^T(\Delta\mathbf{\Sigma}_{ij})\omega}{\omega^T \bar{\mathbf{\Sigma}}_{ij}\omega}\right)^2. 
\end{aligned} \tag{14}$$

Recursively applying the following inequality $\left(\frac{a}{b}\right)^2 + \left(\frac{c}{d}\right)^2 \geq \left(\frac{a+c}{b+d}\right)^2$, $\forall a, c \geq 0; b, d > 0$ to the error bound in equation (14), we have

$$\varepsilon \leq \sum_{i=1}^{c-1} \sum_{j=i+1}^{c} \sqrt{P_i P_j} - \frac{1}{8} \left( \frac{\sum_{i=1}^{c} \sum_{j=1}^{c} (P_i P_j)^{\frac{5}{4}} |\omega^T \Delta\Sigma_{ij} \omega|}{\sum_{i=1}^{c} \sum_{j=1}^{c} P_i P_j \omega^T \bar{\Sigma}_{ij} \omega} \right)^2. \tag{15}$$

Let $\bar{\Sigma} = \sum_{i=1}^{c} P_i \Sigma_i$ be the global covariance matrix. Then we have

$$\sum_{i=1}^{c} \sum_{j=1}^{c} P_i P_j \bar{\Sigma}_{ij} = \frac{1}{2} \sum_{i=1}^{c} \sum_{j=1}^{c} P_i P_j (\Sigma_i + \Sigma_j) = \bar{\Sigma}. \tag{16}$$

Combining equations (15) and (16), we have

$$\varepsilon \leq \sum_{i=1}^{c-1} \sum_{j=i+1}^{c} \sqrt{P_i P_j} - \frac{1}{8} \left( \frac{\sum_{i=1}^{c} \sum_{j=1}^{c} (P_i P_j)^{\frac{5}{4}} |\omega^T \Delta\Sigma_{ij} \omega|}{\omega^T \bar{\Sigma} \omega} \right)^2. \tag{17}$$

Assume that the prior probabilities of the classes are the same, i.e., $P_i = P_j = P$, which holds for most EEG experiments. Then equation (17) becomes

$$\varepsilon \leq \sum_{i=1}^{c-1} \sum_{j=i+1}^{c} P - \frac{1}{8} \left( \frac{P^{\frac{5}{2}} \sum_{i=1}^{c} \sum_{j=1}^{c} |\omega^T (\Sigma_i - \Sigma_j) \omega|}{2 \omega^T \bar{\Sigma} \omega} \right)^2. \tag{18}$$

On the other hand, from $\bar{\Sigma} = \sum_{i=1}^{c} P_i \Sigma_i = \sum_{i=1}^{c} P \Sigma_i$, we obtain that

$$P \sum_{i=1}^{c} |\omega^T (\Sigma_i - \Sigma_j) \omega| \geq \left| \sum_{j=1}^{c} P \omega^T (\Sigma_i - \Sigma_j) \omega \right| = |\omega^T (\Sigma_i - \bar{\Sigma}) \omega|. \tag{19}$$

Combining equations (19) and (18), we obtain that

$$\varepsilon \leq \sum_{i=1}^{c-1} \sum_{j=i+1}^{c} P - \frac{1}{8} \left( \frac{P^{\frac{3}{2}} \sum_{i=1}^{c} |\omega^T (\Sigma_i - \bar{\Sigma}) \omega|}{2 \omega^T \bar{\Sigma} \omega} \right)^2. \tag{20}$$

## 3.2 MCSSPs Based on Multi-class Bayes Error Estimation

Let $\hat{\Sigma}_i$ $(k = 1, \cdots, c)$ denote the new class covariance matrices computed via equation (8). Then to minimize the Bayes error, we should minimize its upper bound, which boils down to maximizing the following discriminant criterion

$$J(\omega) = \frac{\sum_{i=1}^{c} |\omega^T (\hat{\Sigma}_i - \hat{\bar{\Sigma}}) \omega|}{\omega^T \hat{\bar{\Sigma}} \omega}. \tag{21}$$

where $\hat{\bar{\Sigma}}$ is the global covariance matrix. Based on this criterion, we define the $k$ optimal spatial filters of MCSSPs as follows:

$$\omega_1 = \arg \max_{\omega} \frac{\sum_{i=1}^{c} |\omega^T (\hat{\Sigma}_i - \hat{\bar{\Sigma}}) \omega|}{\omega^T \hat{\bar{\Sigma}} \omega},$$
$$\cdots$$
$$\omega_k = \arg \max_{\substack{\omega^T \hat{\bar{\Sigma}} \omega_j = 0, \\ j = 1, \cdots, k-1}} \frac{\sum_{i=1}^{c} |\omega^T (\hat{\Sigma}_i - \hat{\bar{\Sigma}}) \omega|}{\omega^T \hat{\bar{\Sigma}} \omega}. \tag{22}$$

Let $\hat{\tilde{\Sigma}}_i = \hat{\bar{\Sigma}}^{-\frac{1}{2}} \hat{\Sigma}_i \hat{\bar{\Sigma}}^{-\frac{1}{2}}$ $(i = 1, \cdots, c)$ and $\alpha = \hat{\bar{\Sigma}}^{\frac{1}{2}} \omega$. Then solving the optimization problem of equation (22) is equivalent to solving the following optimization problem

$$\alpha_1 = \arg \max_{\alpha} \frac{\sum_{i=1}^{c} |\alpha^T (\hat{\tilde{\Sigma}}_i - \mathbf{I}) \alpha|}{\alpha^T \alpha},$$
$$\cdots$$
$$\alpha_k = \arg \max_{\alpha^T \mathbf{U}_{k-1} = \mathbf{0}} \frac{\sum_{i=1}^{c} |\alpha^T (\hat{\tilde{\Sigma}}_i - \mathbf{I}) \alpha|}{\alpha^T \alpha}, \tag{23}$$

where $\mathbf{U}_{k-1} = [\alpha_1, \cdots, \alpha_{k-1}]$ and $\mathbf{I}$ is the identity matrix. Suppose that $s_i \in \{+1, -1\}$ denotes the positive or negative sign of $\alpha^T(\hat{\boldsymbol{\Sigma}}_i - \mathbf{I})\alpha$. Then

$$|\alpha^T(\hat{\boldsymbol{\Sigma}}_i - \mathbf{I})\alpha| = \alpha^T s_i(\hat{\boldsymbol{\Sigma}}_i - \mathbf{I})\alpha. \tag{24}$$

So equation (23) can be expressed as

$$
\begin{aligned}
\alpha_1 &= \arg\max_{\alpha} \frac{\alpha^T \sum_{i=1}^c s_i(\hat{\boldsymbol{\Sigma}}_i - \mathbf{I})\alpha}{\alpha^T \alpha}, \\
&\cdots \\
\alpha_k &= \arg\max_{\alpha^T \mathbf{U}_{k-1} = \mathbf{0}} \frac{\alpha^T \sum_{i=1}^c s_i(\hat{\boldsymbol{\Sigma}}_i - \mathbf{I})\alpha}{\alpha^T \alpha}.
\end{aligned}
\tag{25}
$$

Let $\mathbf{T}(\mathbf{s}) = \sum_{i=1}^c s_i(\hat{\boldsymbol{\Sigma}}_i - \mathbf{I})$, where $\mathbf{s} = [s_1, s_2, \cdots, s_c]^T$ and $s_i \in \{+1, -1\}$. Then the first vector $\alpha_1$ defined in equation (25) is the principal eigenvector associated with the largest eigenvalue of the matrix $\mathbf{T}(\mathbf{s})$. Suppose that we have obtained the first $k$ vectors $\alpha_1, \cdots, \alpha_k$. To solve the $(k+1)$-th vector $\alpha_{k+1}$, we introduce Theorems 1 and 2 below. The similar proofs of both theorems can be found in [9].

**Theorem 1.** *Let $\mathbf{Q}_k \mathbf{R}_k$ be the QR decomposition of $\mathbf{U}_k$. Then $\alpha_{k+1}$ defined in (25) is the principal eigenvector corresponding to the largest eigenvalue of the following matrix*

$$(\mathbf{I}_d - \mathbf{Q}_k \mathbf{Q}_k^T)\mathbf{T}(\mathbf{s})(\mathbf{I}_d - \mathbf{Q}_k \mathbf{Q}_k^T).$$

**Theorem 2.** *Suppose that $\mathbf{Q}_k \mathbf{R}_k$ is the QR decomposition of $\mathbf{U}_k$. Let $\mathbf{U}_{k+1} = (\mathbf{U}_k \quad \alpha_{k+1})$, $\mathbf{q} = \alpha_{k+1} - \mathbf{Q}_k(\mathbf{Q}_k^T \alpha_{k+1})$, and $\mathbf{Q}_{k+1} = \left(\mathbf{Q}_k \quad \frac{\mathbf{q}}{\|\mathbf{q}\|}\right)$. Then*

$$\mathbf{Q}_{k+1} \begin{pmatrix} \mathbf{R}_k & \mathbf{Q}_k^T \alpha_{k+1} \\ \mathbf{0} & \|\mathbf{q}\| \end{pmatrix}$$

*is the QR decomposition of $\mathbf{U}_{k+1}$.*

The above two theorems are crucial to design our fast algorithm for solving MCSSPs: Theorem 1 makes it possible to use the power method to solve MCSSPs, while Theorem 2 makes it possible to update $\mathbf{Q}_{k+1}$ from $\mathbf{Q}_k$ by adding a single column. Moreover, it is notable that

$$\mathbf{I}_d - \mathbf{Q}_k \mathbf{Q}_k^T = \prod_{i=1}^k (\mathbf{I}_d - \mathbf{q}_i \mathbf{q}_i^T) = (\mathbf{I}_d - \mathbf{Q}_{k-1} \mathbf{Q}_{k-1}^T)(\mathbf{I}_d - \mathbf{q}_k \mathbf{q}_k^T), \tag{26}$$

where $\mathbf{q}_i$ is the $i$-th column of $\mathbf{Q}_k$. Equation (26) makes it possible to update the matrix $(\mathbf{I}_d - \mathbf{Q}_k \mathbf{Q}_k^T)\mathbf{T}(\mathbf{s})(\mathbf{I}_d - \mathbf{Q}_k \mathbf{Q}_k^T)$ from $(\mathbf{I}_d - \mathbf{Q}_{k-1} \mathbf{Q}_{k-1}^T)\mathbf{T}(\mathbf{s})(\mathbf{I}_d - \mathbf{Q}_{k-1} \mathbf{Q}_{k-1}^T)$ by the rank-one update technique.

Let $\mathbf{S} = \{\mathbf{s} | \mathbf{s} \in \{+1, -1\}^c\}$ denote the parameter vector set, whose cardinality is $2^c$. Then we have that

$$\max_{\|\alpha\|=1} \sum_{i=1}^c |\alpha^T(\hat{\boldsymbol{\Sigma}}_i - \mathbf{I})\alpha| = \max_{\mathbf{s} \in \mathbf{S}} \max_{\|\alpha\|=1} \alpha^T \mathbf{T}(\mathbf{s})\alpha. \tag{27}$$

If $c$ is not too large, a full search on $\mathbf{S}$ similar to that proposed in [9] is affordable. We present the pseudo-code of our MCSSPs method using the full search on $\mathbf{S}$ in Algorithm 1. However, if $c$ is a bit large, we may adopt a similar approach as proposed in [10], which is based on a greedy search, to find the suboptimal solution. The pseudo-code based on the greedy search is given in Algorithm 2.

## 4 EEG Feature Extraction Based on the MCSSPs

Let $\mathbf{X}_i^t$ be the EEG sample points from the $t$th trial under the $i$th condition (i.e., the $i$th class). Let $\omega_j$ be the $j$th optimal spatial filter of the MCSSPs method. Construct the new data $\hat{\mathbf{X}}_i^t = \begin{pmatrix} \mathbf{X}_i^t \\ \delta^\tau(\mathbf{X}_i^t) \end{pmatrix}$, and let

$$\hat{\mathbf{p}}_{i,j}^t = \omega_j^T \hat{\mathbf{X}}_i^t \tag{28}$$

**Algorithm 1:** The MCSSPs Algorithm Based on the Full Search Strategy

**Input:**
- Input data matrix $\mathbf{X}$ and the class label vector $l$.

**Initialization:**

1. Compute the average covariance matrices $\hat{\boldsymbol{\Sigma}}_i$ $(i = 1, \cdots, c)$ and $\hat{\bar{\boldsymbol{\Sigma}}}$;

2. Perform SVD of $\hat{\bar{\boldsymbol{\Sigma}}}$: $\hat{\bar{\boldsymbol{\Sigma}}} = \mathbf{U}\boldsymbol{\Lambda}\mathbf{U}^T$, compute $\hat{\bar{\boldsymbol{\Sigma}}}^{-\frac{1}{2}} = \mathbf{U}\boldsymbol{\Lambda}^{-\frac{1}{2}}\mathbf{U}^T$ and $\hat{\bar{\boldsymbol{\Sigma}}}^{-1} = \mathbf{U}\boldsymbol{\Lambda}^{-1}\mathbf{U}^T$;

3. Compute $\hat{\bar{\boldsymbol{\Sigma}}}_i = \hat{\bar{\boldsymbol{\Sigma}}}^{-\frac{1}{2}}\hat{\boldsymbol{\Sigma}}_i\hat{\bar{\boldsymbol{\Sigma}}}^{-\frac{1}{2}}$ and $\Delta\hat{\bar{\boldsymbol{\Sigma}}}_i = \hat{\bar{\boldsymbol{\Sigma}}}_i - \mathbf{I}$ $(i = 1, \cdots, c)$;

4. Enumerate all the elements of $\mathbf{S}$ and denote them by $\mathbf{S} = \{\mathbf{s}_1, \mathbf{s}_2, \cdots, \mathbf{s}_{2^c}\}$;

**For** $i = 1, 2, \cdots, k$, **Do**

1. For $j$=1 to $2^c$
   - Compute $\mathbf{T}(\mathbf{s}_i)$;
   - Solve the principal eigenvector of $\mathbf{T}(\mathbf{s}_i)\alpha^{(j)} = \lambda^{(j)}\alpha^{(j)}$ via the power iteration method;

2. Select the eigenvector $\alpha$ with the largest eigenvalue $\max_{j=1,\cdots,2^c}\{\lambda^{(j)}\}$;

3. If $i = 1$, then $\mathbf{q}_i \leftarrow \alpha$, $\mathbf{q}_i \leftarrow \mathbf{q}_i/\|\mathbf{q}_i\|$, and $\mathbf{Q}_1 \leftarrow \mathbf{q}_i$;
   else $\mathbf{q}_i \leftarrow \alpha - \mathbf{Q}_{i-1}(\mathbf{Q}_{i-1}^T\alpha)$, $\mathbf{q}_i \leftarrow \mathbf{q}_i/\|\mathbf{q}_i\|$, and $\mathbf{Q}_i \leftarrow (\mathbf{Q}_{i-1} \quad \mathbf{q}_i)$;

4. Compute $\Delta\hat{\bar{\boldsymbol{\Sigma}}}_p \leftarrow \Delta\hat{\bar{\boldsymbol{\Sigma}}}_p - (\Delta\hat{\bar{\boldsymbol{\Sigma}}}_p\mathbf{q}_i)\mathbf{q}_i^T - \mathbf{q}_i(\mathbf{q}_i^T\Delta\hat{\bar{\boldsymbol{\Sigma}}}_p) + \mathbf{q}_i(\mathbf{q}_i^T\Delta\hat{\bar{\boldsymbol{\Sigma}}}_p\mathbf{q}_i)\mathbf{q}_i^T$ $(p = 1, \cdots, c)$;

**Output:** $\omega_i = \hat{\bar{\boldsymbol{\Sigma}}}^{-\frac{1}{2}}\alpha_i$, $i = 1, \cdots, k$.

---

**Algorithm 2:** The MCSSPs Algorithm Based on the Greedy Search Strategy

**Input:**
- Input data matrix $\mathbf{X}$ and the class label vector $l$.

**Initialization:**

1. Compute the average covariance matrices $\hat{\boldsymbol{\Sigma}}_i$ $(i = 1, \cdots, c)$ and $\hat{\bar{\boldsymbol{\Sigma}}}$;

2. Perform SVD of $\hat{\bar{\boldsymbol{\Sigma}}}$: $\hat{\bar{\boldsymbol{\Sigma}}} = \mathbf{U}\boldsymbol{\Lambda}\mathbf{U}^T$, compute $\hat{\bar{\boldsymbol{\Sigma}}}^{-\frac{1}{2}} = \mathbf{U}\boldsymbol{\Lambda}^{-\frac{1}{2}}\mathbf{U}^T$ and $\hat{\bar{\boldsymbol{\Sigma}}}^{-1} = \mathbf{U}\boldsymbol{\Lambda}^{-1}\mathbf{U}^T$;

3. Compute $\hat{\bar{\boldsymbol{\Sigma}}}_i = \hat{\bar{\boldsymbol{\Sigma}}}^{-\frac{1}{2}}\hat{\boldsymbol{\Sigma}}_i\hat{\bar{\boldsymbol{\Sigma}}}^{-\frac{1}{2}}$ and $\Delta\hat{\bar{\boldsymbol{\Sigma}}}_i = \hat{\bar{\boldsymbol{\Sigma}}}_i - \mathbf{I}$ $(i = 1, \cdots, c)$;

**For** $i = 1, 2, \cdots, k$, **Do**

1. Set $\mathbf{s} \leftarrow (1, \cdots, 1)^T$, $\mathbf{s}_1 \leftarrow -\mathbf{s}$, and compute $\mathbf{T}(\mathbf{s})$;

2. Solve the principal eigenvector of $\mathbf{T}(\mathbf{s})\alpha = \lambda\alpha$ associated with the largest absolute eigenvalue $|\lambda|$ via the power iteration method. Set $\lambda_0 \leftarrow |\lambda|$;
   **While** $\mathbf{s} \neq \mathbf{s}_1$, **Do**

   (a) Set $\mathbf{s}_1 \leftarrow \mathbf{s}$;

   (b) For $j = 1, 2, \cdots, c$, Do
      - Set $s_j \leftarrow -s_j$, where $s_j$ denotes the $j$th element of $\mathbf{s}$. Compute $\mathbf{T}(\mathbf{s})$;
      - Solve the principal eigenvector of $\mathbf{T}(\mathbf{s})\alpha = \lambda\alpha$ associated with the largest absolute eigenvalue $|\lambda|$ via the power iteration method, and set $\lambda_1 \leftarrow |\lambda|$;
      - If $\lambda_1 \leq \lambda_0$, then $s_j \leftarrow -s_j$, else $\lambda_0 \leftarrow \lambda_1$;

   (c) Compute $\mathbf{T}(\mathbf{s})$ and solve the principal eigenvector $\alpha_i$ of $\mathbf{T}(\mathbf{s})\alpha_i = \lambda\alpha_i$ associated with the largest absolute eigenvalue $|\lambda|$ via the power iteration method;

3. If $i = 1$, then $\mathbf{q}_i \leftarrow \alpha_i$, $\mathbf{q}_i \leftarrow \mathbf{q}_i/\|\mathbf{q}_i\|$, and $\mathbf{Q}_1 \leftarrow \mathbf{q}_i$;
   else $\mathbf{q}_i \leftarrow \alpha_i - \mathbf{Q}_{i-1}(\mathbf{Q}_{i-1}^T\alpha_i)$, $\mathbf{q}_i \leftarrow \mathbf{q}_i/\|\mathbf{q}_i\|$, and $\mathbf{Q}_i \leftarrow (\mathbf{Q}_{i-1} \quad \mathbf{q}_i)$;

4. Compute $\Delta\hat{\bar{\boldsymbol{\Sigma}}}_p \leftarrow \Delta\hat{\bar{\boldsymbol{\Sigma}}}_p - (\Delta\hat{\bar{\boldsymbol{\Sigma}}}_p\mathbf{q}_i)\mathbf{q}_i^T - \mathbf{q}_i(\mathbf{q}_i^T\Delta\hat{\bar{\boldsymbol{\Sigma}}}_p) + \mathbf{q}_i(\mathbf{q}_i^T\Delta\hat{\bar{\boldsymbol{\Sigma}}}_p\mathbf{q}_i)\mathbf{q}_i^T$ $(p = 1, \cdots, c)$;

**Output:** $\omega_i = \hat{\bar{\boldsymbol{\Sigma}}}^{-\frac{1}{2}}\alpha_i$, $i = 1, \cdots, k$.

be the projections of the EEG data $\hat{\mathbf{X}}^t$ onto the projection vector $\omega_j$. Then the covariance of the elements in the projections $\hat{\mathbf{p}}_{i,j}^t$ can be expressed as

$$v_{i,j}^t = \text{var}(\omega_j^T \hat{\mathbf{X}}_i^t) = \omega_j^T \hat{\mathbf{\Sigma}}_i^t \omega_j. \tag{29}$$

where $\hat{\mathbf{\Sigma}}_i^t$ denotes the covariance matrix of the EEG data in the $t$th trial of the $i$th class.

For all the $k$ spatio-spectral filters $\omega_1, \cdots, \omega_k$, we obtain the $k$ features $v_{i,j}^t$ ($j = 1, \cdots, k$) from the $t$th trial of EEG data. Now let $\mathbf{v}_i^t = [v_{i,1}^t, \cdots, v_{i,k}^t]^T$ be the feature vector associated with the $t$th trial of the $i$th class. Similar to the method used in [2], the following log-transformation form is used as the final feature vector of the EEG signal:

$$\mathbf{f}_i^t = \log\left(\frac{\mathbf{v}_i^t}{\sum_k v_{i,k}^t}\right), \tag{30}$$

where the log function is applied to each element of the vector independently. The log-transformation serves to approximate the normal distribution of the data [2].

For the given unknown EEG data $\mathbf{Z}$, we use the same procedures to extract the corresponding features, i.e., we first construct the new data $\hat{\mathbf{Z}} = \begin{pmatrix} \mathbf{Z} \\ \delta^\tau(\mathbf{Z}) \end{pmatrix}$, and then adopt the above method to extract the corresponding discriminant feature vector $\mathbf{f}^z$, where

$$\mathbf{f}^z = \log\left(\frac{\mathbf{v}^z}{\sum_k v_k^z}\right), \ \mathbf{v}^z = [v_1^z, \cdots, v_k^z]^T, \ \text{and} \ v_j^z = \omega_j^T \hat{\mathbf{\Sigma}}^z \omega_j, \tag{31}$$

in which $\hat{\mathbf{\Sigma}}^z$ denotes the covariance matrix of $\hat{\mathbf{Z}}$.

After obtaining the discriminant feature vectors $\mathbf{f}_i^t$ ($i = 1, \cdots, c$; $t = 1, 2 \cdots, n_i$) and $\mathbf{f}^z$, we can classify the unknown EEG data into one of the $c$ classes by using a classifier, e.g., the K-nearest neighbor (K-NN) classifier [7].

# 5 Experiments

To test the performance of our MCSSPs method, we use the real world EEG data set to conduct experiments. The data set used here is from "BCI competition 2005" - data set IIIa [11]. This data set consists of recordings from three subjects (k3b, k6b, and l1b), which performed four different motor imagery tasks (left/right hand, one foot, or tongue) according to a cue. During the experiments, the EEG signal is recorded in 60 channels, using the left mastoid as reference and the right mastoid as ground. The EEG was sampled at 250 Hz and was filtered between 1 and 50 Hz with the notch filter on. Each trial lasted for 7 s, with the motor imagery performed during the last 4 s of each trial. For subjects k6b and l1b, a total of 60 trials per condition were recorded. For subject k3b, a total of 90 trials per condition were recorded. Similar to the method in [5], we discard the four trials of subject k6b with missing data. For each trial of the EEG raw data, we only use part of the sample points, i.e., from No.1001 to No.1750, as the experiment data since they carry most of the information in the EEG signal. Consequently, each trial contains 750 data points. We adopt the two-fold cross validation strategy to perform the experiment, i.e., for all the trials of each condition per subject, we divide them into two groups. Each group is used as training data and testing data once. We conduct five rounds of experiments in total, with different divisions of the training and testing data sets, to obtain ten recognition rates, which are averaged as the final recognition rate. For comparison, we also conduct the same experiment using both MCSPs methods proposed by [4] and [5], respectively. To better identify the effect of using different EEG filters, a simple classifier, K-NN classifier with the Euclidean distance and 7 nearest neighbors, is used for final classification.

Table 1 shows the average classification rates (%) versus the standard deviations (%) of the three methods[2], while figure 1 shows the average recognition rates of our MCSSPs method with different choices of the delayed time $\tau$. From table 1, we can see that the MCSSPs method achieves much better classification performance than the MCSPs methods.

Table 1: Comparison of the classification rates (%) versus standard deviations (%) between MCSPs and MCSSPs.

| Subject | MCSPs [4] | MCSPs [5] | MCSSPs/Bayes |
|---------|-----------|-----------|--------------|
| k3b | 46.17 (6.15) | 84.89 (2.74) | **85.83 (2.23)** |
| k6b | 33.54 (4.27) | 50.09 (2.59) | **56.28 (3.87)** |
| l1b | 35.17 (3.92) | 62.08 (3.99) | **68.58 (6.16)** |

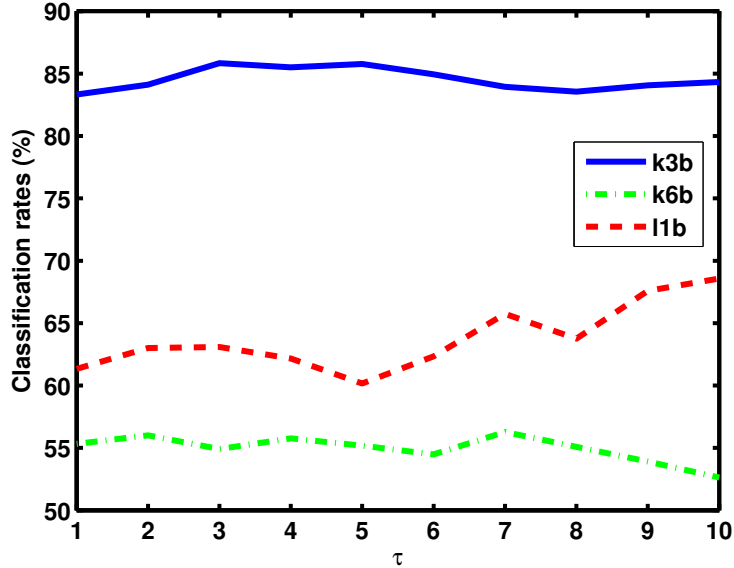

Figure 1: The classification rates (%) of our MCSSPs method with different choices of $\tau$.

## 6  Conclusions

In this paper, we extended the two-class CSSPs method to the multi-class cases via the Bayes error estimation. We first proposed a novel theory on multi-class Bayes error estimation, which has a closed-form solution to find the optimal discriminant vectors for feature extraction. Then we applied the multi-class Bayes error estimation theory to generalize the two-class CSSPs method to multi-class cases. The experiments on the data set IIIa from BCI competition 2005 have shown that our MCSSPs method is superior to the MCSPS methods. With more elaborate treatments, e.g., preprocessing the EEG signal and adopting a more advanced classifier, even higher classification rates are possible. These will be reported in our forthcoming papers.

## Acknowledgment

This work was partly supported by National Natural Science Foundation of China under Grants 60503023 and 60872160.

## Footnotes

[1]This model is often assumed in the literature, e.g., [5].

[2] The results using the MCSPs method proposed in [5] are inferior to those reported in [5] because we did not pre-filter the EEG signals with a Butterworth filter and did not use the logistic regression classifiers for classification either, as we are more interested in comparing the effect of different EEG filters.

## References

[1] B. Blankertz, G. Curio, & K.-R. Müller (2002) Classifying single trial EEG: towards brain computer interfacing. In: T.G. Dietterich, S. Bechker, Z. Ghaharamani (Eds.), *Advances in Neural Information Processing Systems* 14, pp.157-164. Cambridge, MA:MIT Press.

[2] H. Ramoser, J. Mueller-Gerking, & G. Pfurtscheller (2000) Optimal spatial filtering of single trial EEG during imaged hand movement. *IEEE Transactions on Rehabilitation Engineering*. 8(4):441-446.

[3] S. Lemm, B. Blanketz, G. Curio, & K.-R. Müller (2005) Spatio-spectral filters for improved classification of single trial EEG. *IEEE Transactions on Biomedical Engineering*. 52(9):1541-1548.

[4] G. Dornhege, B. Blankertz, G. Curio, & K.-R. Müller (2004) Boosting bit rates in noninvasive EEG single-trial classifications by feature combination and multiclass paradigms. *IEEE Transactions on Biomedical Engineering*. 51(6):993-1002.

[5] M. Grosse-Wentrup, & M. Buss (2008) Multiclass Common Spatial Patterns and Information Theoretic Feature Extraction. *IEEE Transactions on Biomedical Engineering*. 55:1991-2000.

[6] L. C. Parra, C. D. Spence, A. D. Gerson, & P. Sajda (2005) Recipes for linear analysis of EEG. *Neuroimage*, 28:326-341.

[7] K. Fukunaga (1990) Introduction to Statistical Pattern Recognition (Second Edition). *New York: Academic Press*.

[8] J.T. Chu & J.C. Chuen (1967) Error Probability in Decision Functions for Character Recognition. *Journal of the Association for Computing Machinery*. 14(2):273-280.

[9] W. Zheng (2009) Heteroscedastic Feature Extraction for Texture Classification. *IEEE Signal Processing Letters*, 16(9):766-769.

[10] W. Zheng, H. Tang, Z. Lin, & T.S. Huang (2009) A Novel Approach to Expression Recognition from Non-frontal Face Images. *Proceedings of 2009 IEEE International Conference on Computer Vision (ICCV2009)*, pp.1901-1908.

[11] G. Blankertz, K.R. Mueller, D. Krusienski, G. Schalk, J.R. Wolpaw, A. Schloegl, G. Pfurtscheller, J. R. Millan, M. Schroeder, & N. Birbaumer (2006) The BCI competition III: Validating alternative approaches to actual BCI problems. *IEEE Transactions on Rehabilitation Engineering* 14:153-159.

